# Effects of Synaptic Weight Diffusion on Learning in Decision Making Networks

**Kentaro Katahira**[1,2,3], **Kazuo Okanoya**[1,3] **and Masato Okada**[1,2,3]
[1]ERATO Okanoya Emotional Information Project, Japan Science Technology Agency
[2]Graduate School of Frontier Sciences, The University of Tokyo, Kashiwa, Chiba 277-8561, Japan
[3]RIKEN Brain Science Institute, Wako, Saitama 351-0198, Japan
`katahira@mns.k.u-tokyo.ac.jp` `okanoya@brain.riken.jp`
`okada@k.u-tokyo.ac.jp`

## Abstract

When animals repeatedly choose actions from multiple alternatives, they can allocate their choices stochastically depending on past actions and outcomes. It is commonly assumed that this ability is achieved by modifications in synaptic weights related to decision making. Choice behavior has been empirically found to follow Herrnstein's matching law. Loewenstein & Seung (2006) demonstrated that matching behavior is a steady state of learning in neural networks if the synaptic weights change proportionally to the covariance between reward and neural activities. However, their proof did not take into account the change in entire synaptic distributions. In this study, we show that matching behavior is not necessarily a steady state of the covariance-based learning rule when the synaptic strength is sufficiently strong so that the fluctuations in input from individual sensory neurons influence the net input to output neurons. This is caused by the increasing variance in the input potential due to the diffusion of synaptic weights. This effect causes an undermatching phenomenon, which has been observed in many behavioral experiments. We suggest that the synaptic diffusion effects provide a robust neural mechanism for stochastic choice behavior.

## 1 Introduction

Decision making has often been studied in experiments in which a subject repeatedly chooses actions and rewards are given depending on the action. The choice behavior of subjects in such experiments is known to obey Herrnstein's matching law [1]. This law states that the proportional allocation of choices matches the relative reinforcement obtained from those choices. The neural correlates of matching behavior have been investigated [2] and the computational models that explain them have been developed [3, 4, 5, 6, 7]

Previous studies have shown that the learning rule in which the weight update is made proportionally to the covariance between reward and neural activities lead to matching behavior (we simply refer to this learning rule as the covariance rule) [3, 7]. In this study, by means of a statistical mechanical approach [8, 9, 10, 11], we analyze the properties of the covariance rule in a limit where the number of plastic synapses is infinite . We demonstrate that matching behavior is not a steady state of the covariance rule under three conditions: (1) learning is achieved through the modification of the synaptic weights from sensory neurons to the value-encoding neurons; (2) individual fluctuations in sensory input neurons are so large that they can affect the potential of value-coding neurons (possibly via sufficiently strong synapses); (3) the number of plastic synapses that are involved in learning is large. This result is caused by the diffusion of synaptic weights. The term "diffusion" refers to a phenomenon where the distributions over the population of synaptic weights broadens. This diffusion increases the variance in the potential of output units since the broader synaptic weight distributions are, the more they amplify fluctuations in individual inputs. This makes the choice

behavior of the network more random and moves the probabilities of choosing alternatives to equal probabilities, than that predicted by the matching law. This outcome corresponds to the under-matching phenomenon, which has been observed in behavioral experiments.

Our results suggest that when we discuss the learning processes in a decision making network, it may be insufficient to only consider a steady state for individual weight updates, and we should therefore consider the dynamics of the weight distribution and the network architecture. This proceeding is a short version of our original paper [12], with the model modified and new results included.

## 2 Matching Law

First, let us formulate the matching law. We will consider a case with two alternatives (each denoted as $A$ and $B$), which has generally been studied in animal experiments. Here, we consider stochastic choice behavior, where at each time step, a subject chooses alternative $a$ with probability $p_a$. We denote the reward as $r$. For the sake of simplicity, we restrict $r$ to a binary variable: $r = 0$ represents the absence of a reward, and $r = 1$ means that a reward is given. The expected return, $\langle r|a \rangle$, refers to the average reward per choice $a$, and the income, $I_a$, refers to the total amount of reward resulting from the choice $a$ and $I_a / \left( \sum_{a'}^{n_a} I_{a'} \right)$ is a fractional income from choice $a$. For a large number of trials, this equals $\langle r|a \rangle p_a$. $\langle r \rangle = \sum_{a'}^{n_a} \langle r|a' \rangle p_{a'}$ is an average reward per trial over possible choice behavior. The matching law states that $I_a / \left( \sum_{a'}^{n_a} I_{a'} \right) = p_a$ for all $a$ with $p_a \neq 0$. For a large number of trials, the fraction of income from an alternative $a$ is expressed as $\frac{\langle r|a \rangle p_a}{\sum_{a'} \langle r|a' \rangle p_{a'}} = \frac{\langle r|a \rangle p_a}{\langle r \rangle}$ Then, the matching law states that this quantity equals $p_a$ for all $a$. To make this hold, it should satisfy

$$\langle r|A \rangle = \langle r|B \rangle = \langle r \rangle, \tag{1}$$

if $p_A \neq 0$ and $p_B \neq 0$. Note that $\langle r|a \rangle$ is the average reward given the current choice, and this is a function of the past choice. Equation 1 is a condition for the matching law, and we will often use this identity.

## 3 Model

**Decision Making Network:** The decision making network we study consists of sensory-input neurons and output neurons that represent the subjective value of each alternative (we call the output neurons value-encoding neurons). The network is divided into two groups ($A$ and $B$), which participate in choosing each alternative. Sensory cues from both targets are given simultaneously via the $N$-neuron population, $\boldsymbol{x}^A = (x_1^A, ..., x_N^A)$ and $\boldsymbol{x}^B = (x_1^B, ..., x_N^B)$ [1] Each component of input vectors $\boldsymbol{x}^A$ and $\boldsymbol{x}^B$ independently obeys a gaussian distribution with mean $X_0$ and variance one (these quantities can be spike counts during stimulus presentation).

The choice is made in such a way that alternative $a$ is chosen if the potential of output unit $u^a$, which will be specified below, is higher than that of the other alternative. Although we do not model this comparison process explicitly, it can be carried out via a winner-take-all competition mediated by feedback inhibition, as has been commonly assumed in decision making networks [3, 13]. In this competition, the "winner" group gains a high firing rate while the "loser" enters a low firing state [13]. Let $y^A$ and $y^B$ denote the final output of an output neuron after competition and this is determined as

$$y^A = 1, \ y^B = 0, \ \text{if} \ u^A \geq u^B,$$
$$y^A = 0, \ y^B = 1, \ \text{if} \ u^A < u^B.$$

With the synaptic efficacies (or weights) $\boldsymbol{J}^A = (J_1^A, ..., J_N^A)$ and $\boldsymbol{J}^B = (J_1^B, ..., J_N^B)$, the net input to the output units are given by

$$h^a = \sum_{i=1}^N J_i^a x_i^a, \ \ a = A, B. \tag{2}$$

We assume that $J_i^a$ is scaled as $O(1/\sqrt{N})$. This means that the mean of $h^a$ is $O(\sqrt{N})$, thus diverges for large $N$, while the variance is kept of order unity. This is a key assumption of our models. If $J_i^a$ is scaled as $O(1/N)$ instead, the individual fluctuations in $x_i^a$ are averaged out. It has been shown that the mean of the potential are kept of order unity while fluctuations in external sources ($x_i^a$) that are of order unity affect the potential in output neuron, under the condition that recurrent inputs from inhibitory interneurons, excitatory recurrent inputs, and input from external sources ($x_i^a$) are balanced [14]. We do not explicitly model this recurrent balancing mechanism, but phenomenologically incorporate it as follows.

Using the order parameters

$$l_a = ||\boldsymbol{J}^a||, \;\; \bar{J}_a = \frac{1}{\sqrt{N}} \sum_{i=1}^{N} J_i^a, \tag{3}$$

we find $h^a \sim \mathcal{N}(\sqrt{N} X_0 \bar{J}_a, l_a^2)$ where $\mathcal{N}(\mu, \sigma^2)$ denotes the gaussian distribution with mean $\mu$ and variance $\sigma^2$. We assume $u_a$ obeys a gaussian distribution of mean $C_a u^a / \sqrt{N}$, and variance $C_a \mathrm{Var}[u^a] + \sigma_p^2$ due to the reccurent balancing mechanism [14]. $C_A$, $C_B$ and $\sigma_p^2$ are constants that are determined according to the specific model architecture of reccurent network, but we set $C_A = C_B = 1$ since they do not affect the qualitative properties of the model. Then, $u^a$ is computed as $u^a = h^a - \bar{h}_{\mathrm{rec}}^a + \sigma_p \epsilon$ with $\bar{h}_{\mathrm{rec}}^a = (1 - 1/\sqrt{N}) E[h^a]$ where $E[h^a] = \sqrt{N} X_0 \bar{J}_a$ and $\epsilon$ is a gaussian random variable with unit mean and unit variance. Then, $u_a$ obey the independent Gaussian distributions whose means and variances are respectively given by $\bar{J}_a$ and $l_a^2 + \sigma_p^2$. From this, the probability that the network will choose alternative $A$ can be described as

$$p_A = \frac{1}{2} \mathrm{erfc} \left\{ -\frac{X_0 (\bar{J}_A - \bar{J}_B)}{\sqrt{2(l_A^2 + l_B^2 + 2\sigma_p^2)}} \right\}. \tag{4}$$

where $\mathrm{erfc}(\cdot)$ is the complementary error function, $\mathrm{erfc}(x) = \frac{2}{\sqrt{\pi}} \int_x^\infty e^{-t^2} dt$. This expression is in a closed form of the order parameters. Thus, if we can describe the evolution of these order parameters, we can completely describe how the behavior of the model changes as a consequence of learning. In the following, we will often use an additional order parameter, the variance of weight, $\sigma_a^2$. This parameter is more convenient for gaining insights into the evolution of the weight than the weight norm, $l_a$. The diffusion of weight distributions is reflected by increases in $\sigma_a^2$, i.e., the differences between the growth of the second order moment of weight distribution $l_a^2$ and that of the square of its mean $\bar{J}_a^2$.

**Learning Rules:** We consider following two learning rules that belong to the class of the covariance learning rule:

Reward-modulated (RM) Hebb rule:

$$J_i^a(t+1) = J_i^a(t) + \frac{\eta}{N} \left[ r(t) - \bar{r}(t) \right] y^a(t)(x_i^a(t) - c_x), \tag{5}$$

Delta rule:

$$J_i^a(t+1) = J_i^a(t) + \frac{\eta}{N} \left[ r(t) - \bar{r}(t) \right] (x_i^a(t) - c_x), \tag{6}$$

where $\eta$ is the learning rate, $\bar{\cdot}$ denotes the expected value and $c_x$ is a constant. The expectation of these updates is proportional to covariance between the reward, $r$, and a measure of neural activity ($y^a(x_i^a - c_x)$ for RM-Hebb rule, and $x_i^a - c_x$ for the delta rule). Variants of the RM-Hebb rule have recently been studied intensively [4, 15, 16, 17, 18, 19, 20]. The delta rule has been used as an example of the covariance rule [3, 7] and has also been used for the learning rule in the model of perceptual learning [21]. The expected reward, $\bar{r}$, can be estimated, e.g., with an exponential kernel such as $\bar{r}(t+1) = (1 - \gamma)r(t) + \gamma \bar{r}(t)$ with a constant $\gamma$. We assume that $c_x = (1 - 1/\sqrt{N})X_0$ to simplify the following analysis [2].

# 4 Macroscopic Description of Learning Processes

Here, following the statistical mechanical analysis of on-line learning [8, 9, 10, 11], we derive equations that describe the evolution of the order parameters. To do this, we first rewrite the learning rule in a vector form:

$$\boldsymbol{J}^a(t+1) = \boldsymbol{J}^a(t) + \frac{1}{N} F_a \left(\boldsymbol{x}^a - \boldsymbol{c}_x\right), \qquad (7)$$

where for the RM-Hebb rule, $F_a = \eta(r_t - \bar{r}_t)y^a$ and for the delta rule, $F_a = \eta(r_t - \bar{r}_t)$. Taking the square norm of each side of equation 7, we obtain $l_a(t+1)^2 = l_a(t)^2 + \frac{2}{N} F_a(t)\,\tilde{h}^a + \frac{1}{N} F_a(t)^2 + O(1/N^2)$, where we have defined $\tilde{h}^a = \sum_{i=1}^{N} J_i^a(x_i^a - c_x)$. Summing up over all components on both sides of equation 7, we obtain $\bar{J}_a(t+1) = \bar{J}_a(t) + \frac{1}{N} F_a(t)\tilde{\boldsymbol{x}}_a$, where we have defined $\tilde{\boldsymbol{x}}_a = \sum_{i=1}^{N}(x_i^a - c_x)$. In both these equations, the magnitude of each update is of order $1/N$. Hence, to change the order parameters of order one, $O(N)$ updates are needed. Within this short period that spans the $O(N)$ updates, the weight change in $O(1/N)$ can be neglected, and the self-averaging property holds. By using this property and introducing continuous "time" scaled by $N$, i.e., $\alpha = t/N$, the evolutions of the order parameters obey ordinary differential equations:

$$\frac{dl_a^2}{d\alpha} = 2\langle F_a \tilde{h}^a \rangle + \langle F_a^2 \rangle, \quad \frac{d\bar{J}_a}{d\alpha} = \langle F_a \tilde{\boldsymbol{x}}_a \rangle, \qquad (8)$$

where $\langle \cdot \rangle$ denotes the ensemble average over all possible inputs and arrivals of rewards. The specific form of the ensemble averages are obtained for reward-dependent Hebbian learning as

$$\langle F_a \tilde{h}^a \rangle = \eta\, p_a \left\{ \langle r|a \rangle - \langle r \rangle \right\} \langle \tilde{h}^a|a \rangle,$$
$$\langle F_a^2 \rangle = \eta^2 p_a \left\{ (1 - 2\langle r \rangle)\langle r|a \rangle + (\langle r \rangle)^2 \right\},$$
$$\langle F_a \tilde{\boldsymbol{x}}_a \rangle = \eta\, p_a \left\{ \langle r|a \rangle - \langle r \rangle \right\} \langle \tilde{\boldsymbol{x}}_a|a \rangle,$$

and for the delta rule,

$$\langle F_a \tilde{h}^a \rangle = \eta \left\{ p_a(\langle r|a \rangle - \langle r|a' \rangle)\langle \tilde{h}^a|a \rangle + (\langle r|a' \rangle - \langle r \rangle)\bar{J}_a \right\},$$
$$\langle F_a^2 \rangle = \eta^2 \left\{ \langle r \rangle(1 - \langle r \rangle) \right\},$$
$$\langle F_a \tilde{\boldsymbol{x}}_a \rangle = \eta \left\{ p_a (\langle r|a \rangle - \langle r|a' \rangle)\langle \tilde{\boldsymbol{x}}_a|a \rangle + \langle r|a' \rangle - \langle r \rangle \right\}.$$

The conditional averages $\langle \tilde{h}^a|a \rangle$ and $\langle \tilde{\boldsymbol{x}}_a|a \rangle$ in these equations are computed as

$$\langle \tilde{h}^a|a \rangle = \bar{J}_a X_0 + \frac{l_a^2}{p_a \sqrt{2\pi L^2}} \exp\left( -\frac{X_0^2 D_{\bar{J}}^2}{2L^2} \right), \quad \langle \tilde{\boldsymbol{x}}_a|a \rangle = X_0 + \frac{\bar{J}_a}{p_a \sqrt{2\pi L^2}} \exp\left( -\frac{X_0^2 D_{\bar{J}}^2}{2L^2} \right), \qquad (9)$$

where we have defined $L = \sqrt{l_A^2 + l_B^2 + 2\sigma_p^2}$ and $D_{\bar{J}} = \bar{J}_B - \bar{J}_A$. The details on the derivation are given in the supplementary material and [12].

Next, we consider weight normalization in which the total length of the weight vector is kept constant. We adopted this weight normalization because of analytical convenience rather than taking biological realism into account. Other weight constraints would produce no clear differences in the following results. Specifically, we constrained the norm of the weight as $||\boldsymbol{J}||^2 = 2$, where $\boldsymbol{J} = (J_1^A, ..., J_N^A, J_1^B, ..., J_N^B)$. This is equivalent to keeping $l_A^2 + l_B^2 = 2$. This is achieved by modifying the learning rule in the following way [22]:

$$\boldsymbol{J}^a(t+1) = \frac{\sqrt{2}(\boldsymbol{J}^a(t) + \frac{1}{N} F_a \boldsymbol{x}^a)}{\sqrt{||\boldsymbol{J}^A(t) + \frac{1}{N} F_A \boldsymbol{x}^A||^2 + ||\boldsymbol{J}^B(t) + \frac{1}{N} F_B \boldsymbol{x}^B||^2}} = \frac{\boldsymbol{J}^a(t) + F_a \boldsymbol{x}^a}{\sqrt{1 + \mathcal{F}/N}}, \qquad (10)$$

with $\mathcal{F} \equiv F_A u^A + F_B u^B + \frac{1}{2}(F_A^2 + F_B^2)$, provided that $||\boldsymbol{J}||^2 = 2$ holds at trial $t$. Expanding the right-hand side to first order in $1/N$, we can obtain the differential equations similarly to Equation 8:

$$\frac{dl_a^2}{d\alpha} = 2\langle F_a \tilde{h}^a \rangle + \langle F_a^2 \rangle - \langle \mathcal{F} \rangle l_a^2, \quad \frac{d\bar{J}_a}{d\alpha} = \langle F_a \tilde{\boldsymbol{x}}_a \rangle - \frac{1}{2}\langle \mathcal{F} \rangle \bar{J}_a. \qquad (11)$$

With $\langle \mathcal{F} \rangle = \langle F_A u^A \rangle + \langle F_B u^B \rangle + \frac{1}{2}(\langle F_A^2 \rangle + \langle F_B^2 \rangle)$, we can find that $d(l_A^2 + l_B^2)/d\alpha$ becomes zero when $l_A^2 + l_B^2 = 2$; thus, the length of the weight is kept constant.

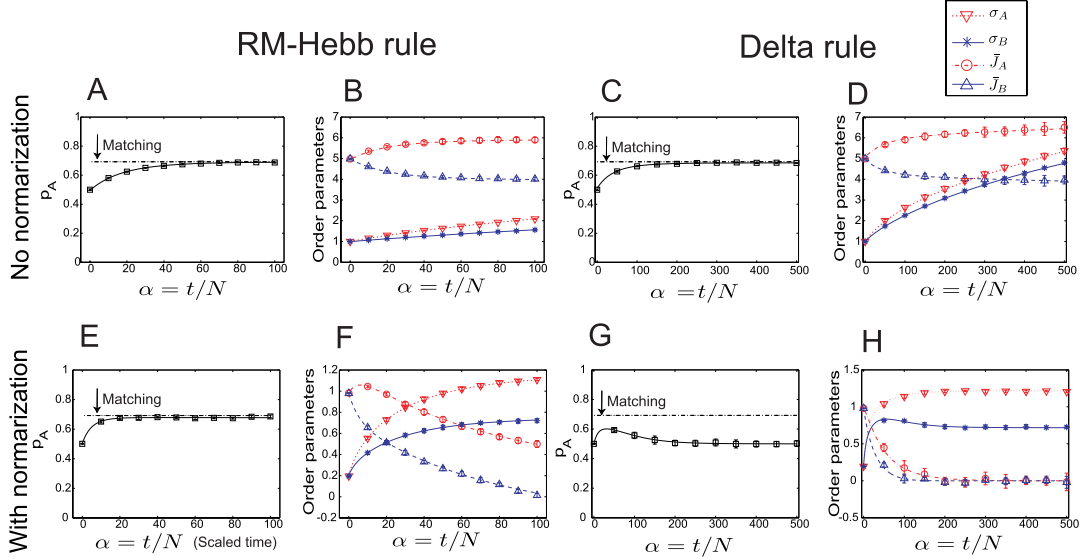

Figure 1: Evolution of choice probability and order parameters for RM-Hebb rules (A, B, E, F) and delta rule (C, D, G, H), without weight normalization (A-D) and with normalization (E-H). Parameters were $X_0 = 2$, $\eta = 0.1$ and $\sigma_p = 1$, and the reward schedule was a VI schedule (see main text) with $\lambda_A = 0.2, \lambda_B = 0.1$. Lines represent results of theory and symbols plot mean of ten trials with computer simulation. Simulations were done for $N = 1,000$. Error bars indicate standard deviation (s.d.). Error bars are almost invisible for choice probability since s.d. is very small.

## 5 Results

To demonstrate the behavior of the model, we used a time-discrete version of a variable-interval (VI) reward schedule, which is commonly used for studying the matching law. In a VI schedule, a reward is assigned to two alternatives stochastically and independently, with a constant probability, $\lambda_a$ for alternative $a$ ($a = A, B$). The reward remains until it is harvested by choosing the alternative. Here, we use $\lambda_A = 0.2, \lambda_B = 0.1$. For this task setting, the choice probability that yields matching behavior (denoted as $p_A^{\text{match}}$) is $p_A^{\text{match}} = 0.6923$. Figure 1(A-D) plots the evolution of choice probability and order parameters in two learning rules without a weight normalization constraint. The lines represent the results for theory and the symbols plot the results for simulations. The results for theory agree well with those for the computer simulations ($N = 1,000$), indicating the validity of our theory. We can see that the choice probability approaches a value that yields matching behavior ($p_A^{\text{match}}$), while the order parameters $\bar{J}_a$ and $\sigma_a$ continue to change without becoming saturated. The weight standard deviation, $\sigma_a$, always increases (the synaptic weight diffusion).

Figure 1(E-H) plots the results with weight normalization. Again, the results for theory agree well with those for computer simulations. For the RM-Hebb rule, the choice probability saturates at a value below $p_A^{\text{match}}$. For the delta rule, the choice probability first approaches $p_A^{\text{match}}$, but without reaching $p_A^{\text{match}}$. It then returns to the uniform choice probability ($p_A = 0.5$) due to its larger diffusion effect than that of the RM-Hebb rule.

### 5.1 Matching Behavior Is Not Necessarily Steady State of Learning

From Figure 1, the choice probability seems to asymptotically approach matching behavior for the case without wight normalization. However, matching behavior is not necessarily a steady state of learning. In Figure 2, the order parameters are initialized so that $p_A(0) = p_A^{\text{match}}$ and then Equations 8 and 11 are numerically solved. We see that $p_A$ does not remain at $p_A^{\text{match}}$ but changes toward the uniform choice ($p_A = 0.5$) for both learning rules. Then, for the RM-Hebb rule, $p_A$ evolves toward $p_A^{\text{match}}$, but not do so for the delta rule. To understand the mechanism for this

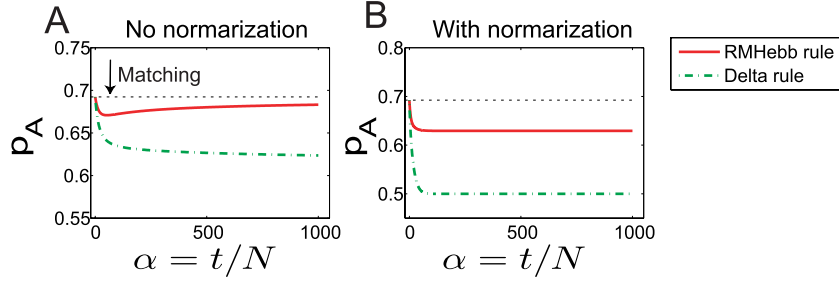

Figure 2: Strict matching behavior is not equilibrium point. We set initial value of order parameters to derive perfect matching for (A) no normalization condition and (B) normalization condition. In both cases, choice probability that yields perfect matching is repulsive. For no normalization condition, initial conditions were first set at $\bar{J}_B = 1.0$, $\sigma_A = \sigma_B = 1.0$ and then $\bar{J}_A$ was determined so that $p_A = p_A^{\text{match}}$. For normalization condition, these values were rescaled so that normalization condition was met.

repulsive property of matching behavior, let us substitute the condition of the matching law, $\langle r|A \rangle = \langle r|B \rangle = \langle r \rangle$ into Equations 11, for the no normalization condition. We then find that $\langle F_a \tilde{h}^a \rangle$ and $\langle F_a \tilde{x}_a \rangle$ are zero but $\langle F_a^2 \rangle$ is non-zero and positive except for the non-interesting case where $r$ always takes the same value. Therefore, when $p_A = p_A^{\text{match}}$, the variance in the weight increases, i.e., $d\sigma_a^2/d\alpha = d(l_a^2 - \bar{J}_a^2)/d\alpha > 0$. This moves the choice probabilities toward unbiased choice behavior, $p_A = 0.5$ (see Equation 4). This is the reason that $p_A^{\text{match}}$ is repulsive. This result is in contrast with the $N = 1$ case [7] where the average changes stop when $p_A$ converges to $p_A^{\text{match}}$.

With weight normalization, $\sqrt{2(l_A^2 + l_B^2)}$ in Equation 4 is always two; thus, the only factor that determines choice probability is the difference between $\bar{J}_A$ and $\bar{J}_B$. Substituting $\langle r|a \rangle = \langle r \rangle, \forall_a$ into Equation 11, only term $\langle F_a^2 \rangle$ remains, and we obtain $d(\bar{J}_B - \bar{J}_A)/(d\alpha) = -\frac{1}{2}(\langle F_A^2 \rangle + \langle F_B^2 \rangle)(\bar{J}_B - \bar{J}_A)$ Except for uninteresting cases where $r$ is always 0 or 1, $\langle F_A^2 \rangle + \langle F_B^2 \rangle > 0$ holds; thus, the absolute difference, $|\bar{J}_B - \bar{J}_A|$, always decreases. Hence, again, the choice probability at $p_A^{\text{match}}$ approaches unbiased choice behavior due to the diffusion effect.

Nevertheless, the choice probability of the RM-Hebb rule without weight normalization asymptotically converges to $p_A^{\text{match}}$. The reason for this can be explained as follows. First, we rewrite the choice probability as

$$p_A = \frac{1}{2}\text{erfc}\left\{ -\frac{X_0(\bar{J}_A - \bar{J}_B)}{\sqrt{2(\bar{J}_A^2 + \bar{J}_B^2 + \sigma_A^2 + \sigma_B^2 + 2\sigma_p^2)}} \right\}. \tag{12}$$

From this expression, we find that the larger the magnitude of $\bar{J}_a$ is, the weaker the effect of increases in $\sigma_a$. The "diffusion term", $\langle F_a^2 \rangle$, which moves $p_A$ away from $p_A^{\text{match}}$ depends on $p_A$ but not on the magnitude of $\bar{J}_a$'s. Thus, within the order parameter set satisfying $p_A = p_A^{\text{match}}$, the larger the magnitudes of $J_a$'s are, the weaker is the repulsive effect. If $|\bar{J}_B - \bar{J}_A| \to \infty$ while $\sigma_A, \sigma_B$ are finite, $p_A$ stays at $p_A^{\text{match}}$. Because $|\bar{J}_B - \bar{J}_A|$ can increase faster than $\sigma_A$ and $\sigma_B$ in the RM-Hebb rule without any weight constraints, the network approaches such situations. This is the reason that in Figure 2A the $p_A$ returned to $p_A^{\text{match}}$ after it was repulsed from $p_A^{\text{match}}$. When weight normalization is imposed, the magnitude of $\bar{J}_a$'s are limited as $|\bar{J}_B - \bar{J}_A| < 2$. Thus, the diffusion effect prevents $p_A$ from approaching $p_A^{\text{match}}$. In the delta rule, the magnitude of $\bar{J}_a$'s cannot increase independently of $\sigma_a$'s. Thus, $p_A$ saturates before it reaches $p_A^{\text{match}}$, where the increase in $|\bar{J}_B - \bar{J}_A|$ and those in $\sigma_a$'s are balanced.

## 5.2 Learning Rate Dependence of Learning Behavior

Next, we investigate how the learning rate, $\eta$, affects the choice behavior. In the "diffusion term", $\langle F_a^2 \rangle$, is a quadratic term in the learning rate $\eta$. In contrast, only the first order terms of $\eta$ appear

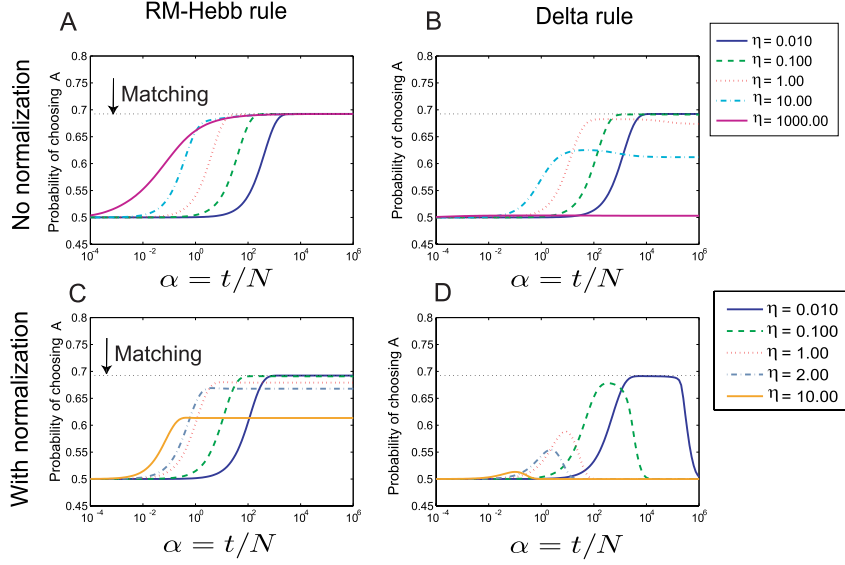

Figure 3: Evolution of choice probability for various learning rates, $\eta$. Top rows are for non-weight normalization condition and bottom rows are for normalization condition. Columns at left are for RM-Hebb rule and those at right are for delta rule. Parameters for model and task schedules are same as those in Figure 1. Initial conditions were set at $\sigma_a = 0.0$, $(a = A, B)$, $\bar{J}_a = 5.0$ for non-normalization condition and $\bar{J}_a = 1.0$ for normalization condition.

in the other terms. Therefore, if $\eta$ is small, the repulsive effect from matching behavior due to the diffusion effect is expected to weaken. Figure 3 plots the dependence of the evolution of $p_A$ on $\eta$. As a whole, as $\eta$ is decreased, the asymptotic value, $p_A$, approaches matching behavior, but relaxation slows down due to the diffusion of synaptic weights. As we previously discussed, the diffusion effect is more evident for the delta rule than for the RM-Hebb rule, and for the weight-normalization condition than the non- normalization condition. This tendency becomes evident as $\eta$ increases.

For the RM-Hebb rule without normalization, networks approach matching behavior even for a very large learning rate ($\eta = 1000$). At the beginning of learning when $\bar{J}_a$ is of small magnitude, the diffusion term, $\langle F_a^2 \rangle$, has a large impact so that it greatly impedes learning for a large $\eta$ case. However, as the magnitude of the differences $\bar{J}_A - \bar{J}_B$ increases, this effect weakens and the dependence of $p_A$ on $\eta$ becomes quite small. Although there is still a deviation from perfect matching (see inset of Figure 3A), the asymptotic value is almost unaffected in the RM-Hebb rule. For the delta rule without normalization, the asymptotic values gradually depend on $\eta$. With normalization constraints, the RM-Hebb rule also demonstrate graded dependence of asymptotic probability on $\eta$. These results reflect the fact that the greater learning rate $\eta$ is, the larger the diffusion effect.

## 5.3 Deviation from Matching Law

Choices by animals in many experiments deviate slightly from matching behavior toward unbiased random choice, a phenomenon called undermatching [2, 23]. The synaptic diffusion effects reproduces this phenomenon. Figure 4A,B plots choice probability for option $A$ as a function of the fraction income from the option. If this function lies at the diagonal line, it corresponds to matching behavior. For the RM-rule with weight normalization, as the learning rate $\eta$ increases, the choice probabilities deviate from matching behavior towards unbiased random choice, $p_A = 0.5$ (Figure 4A). Similar results are obtained for another weight constraint, the hard bound condition (Figure 4B). In this condition, if the updates makes $J_i^a > J_{\max}/\sqrt{N}$ (or $J_i^a < 0$), $J_i^a$ is set to $J_{\max}/\sqrt{N}$ (or 0). We see that the larger the $\eta$ is, the broader the weight distributions due the the synaptic diffusion effects (Figure 4A). This result suggests that the weight diffusion effect causes undermathing regardless of the way of weight constraint, as long as the synaptic weights are confined to a finite range, as predicted by our theory.

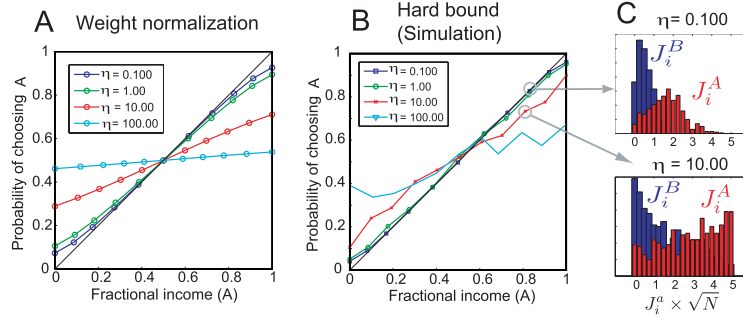

Figure 4: Constraints on synaptic weights leads to the undermatching behavior through synaptic diffusion effects. (A) Choice probability for $A$ as a function of the fraction income for $A$ for the RM-rule with weight normalization. We used VI schedules with $\lambda_A = 0.3a$ and $\lambda_B = 0.3(1 - a)$, varying the constant $a$ ($0 \leq a \leq 1$). The results were obtained using stationaly points of the macroscopic equations. The diagonal line indicates the perfect matching behavior. As the learning rate $\eta$ increases, the choice probabilities deviate from matching behavior towards unbiased random choice, $p_A = 0.5$. (B) The same plot with (A) for the RM-rule with the hard bound condition (the synaptic weights are restricted to the interval $[0, J_{\max}/\sqrt{N}]$ where $J_{\max} = 5.0$) obtained by numerical simulations. Simulations were done for $N = 500$. (C) The weight distribution after convergence for the simulations in (B) indicated by the gray arrows.

## 6 Discussion

In this study, we analyzed the reward-based learning procedure in simple, large-scale decision making networks. To achieve this, we employed techniques from statistical mechanics. Although statistical mechanical analysis has been successively applied to analyze the dynamics of learning the in neural networks, we applied it to reward-modulated learning in decision making networks for the first time, to the best of our knowledge. We have assumed the activities of sensory neurons are independent. In realistic cases, there may be correlations among sensory neurons. The existence of correlation weakens the diffusion effects. However, if there are independent fluctuations, as observed in many physiological studies, the diffusion effects are at play here as well.

If only a single plastic synapse is taken into consideration, covariance learning rules seem to make matching behavior a steady state of learning. However, under certain situations where a large number of synapses simultaneously modify their efficacy, matching behavior cannot be a steady state. This is because the randomness in weight modifications affects the choice probability of the network, and the effect returns to the learning process. These results may offer suggestions for discussing learning behavior in large-scale neural circuits.

Choice behavior in many experiments deviates slightly from matching behavior toward unbiased choice behavior, a phenomenon called undermatching [23, 2]. There are several possible explanations for this phenomenon. The learning rule employed by Soltani & Wang [4] is equivalent to the state-less Q-learning in the literature on reinforcement learning [15]. Sakai & Fukai [5, 6] proved that Q-learning does not lead to matching behavior. Thus, Soltani-Wang's model is intrinsically incapable of reproducing matching behavior. The authors interpreted that the departure from matching behavior due to limitations in the learning rule was a possible mechanism for undermatching. Loewenstein [7] suggested that the mistuning of parameters in the covariance learning rule could cause undermatching. However, we found that in some task settings, the mistuning can cause overmatching, rather than undermatching [12]. Our findings in this study add one possible mechanism for undermatching, i.e., undermatching can be caused by the diffusion of synaptic efficacies. The diffusion effects provide a robust mechanism for undermatching: It reproduces undermatching behavior, regardless of specific task settings.

To achieve random choice behavior, it is thought to require fine-tuning of network parameters [16], whereas random choice behavior is often observed in behavioral experiments. Our results suggest that the broad distributions of synaptic weights observed in experiments [24] can make it easier to realize stochastic random choice behavior perhaps than previously thought.

## Footnotes

[1] This assumption might be the case when the sensory input for each alternative is completely different, e.g., in position, and in color such as those in Sugrue et al.'s experiment [2]. The case that output neurons share the inputs from sensory neurons are analyzed in [12].

[2]From this assumption, this model can be transformed into a simple mathematical equivalent form that the distribution of input $x_i^a$ is replaced with $\mathcal{N}(X_0/\sqrt{N}, 1)$ and the potential in output is replaced with $u^a = \sum_{i=1}^{N} J_i^a x_i^a + \sigma_p \xi^a$, where $\xi^a \sim \mathcal{N}(0, 1)$.

# References

[1] R. J. Herrnstein, H. Rachlin, and D. I. Laibson. *The Matching Law*. Russell Sage Foundation New York, 1997.

[2] L. P. Sugrue, G. S. Corrado, and W. T. Newsome. Matching behavior and the representation of value in the parietal cortex. *Science*, 304(5678):1782–1787, 2004.

[3] Y. Loewenstein and H. S. Seung. Operant matching is a generic outcome of synaptic plasticity based on the covariance between reward and neural activity. *Proceedings of the National Academy of Sciences*, 103(41):15224–15229, 2006.

[4] A. Soltani and X. J. Wang. A biophysically based neural model of matching law behavior: melioration by stochastic synapses. *Journal of Neuroscience*, 26(14):3731–3744, 2006.

[5] Y. Sakai and T. Fukai. The actor-critic learning is behind the matching law: Matching versus optimal behaviors. *Neural Computation*, 20(1):227–251, 2008.

[6] Y. Sakai and T. Fukai. When does reward maximization lead to matching law? *PLoS ONE*, 3(11):e3795, 2008.

[7] Y. Loewenstein. Robustness of learning that is based on covariance-driven synaptic plasticity. *PLoS Computational Biology*, 4(3):e1000007, 2008.

[8] W. Kinzel and P. Rujan. Improving a network generalization ability by selecting examples. *Europhysics Letters*, 13(5):473–477, 1990.

[9] D. Saad. *On-line learning in neural networks*. Cambridge University Press, 1998.

[10] G. Reents and R. Urbanczik. Self-averaging and on-line learning. *Physical Review Letters*, 80(24):5445–5448, 1998.

[11] M. Biehl, N. Caticha, and P. Riegler. Statistical mechanics of on-line learning. *Similarity-Based Clustering*, pages 1–22, 2009.

[12] K. Katahira, K. Okanoya, and M. Okada. Statistical mechanics of reward-modulated learning in decision making networks. under review.

[13] X. J. Wang. Probabilistic decision making by slow reverberation in cortical circuits. *Neuron*, 36(5):955–968, 2002.

[14] C. van Vreeswijk and H. Sompolinsky. Chaotic balanced state in a model of cortical circuits. *Neural Computation*, 10(6):1321–1371, 1998.

[15] A. Soltani, D. Lee, and X. J. Wang. Neural mechanism for stochastic behaviour during a competitive game. *Neural Networks*, 19(8):1075–1090, 2006.

[16] S. Fusi, W. F. Asaad, E. K. Miller, and X. J. Wang. A neural circuit model of flexible sensorimotor mapping: learning and forgetting on multiple timescales. *Neuron*, 54(2):319–333, 2007.

[17] E. M. Izhikevich. Solving the distal reward problem through linkage of STDP and dopamine signaling. *Cerebral Cortex*, 17:2443–2452, 2007.

[18] R. V. Florian. Reinforcement learning through modulation of spike-timing-dependent synaptic plasticity. *Neural Computation*, 19(6):1468–1502, 2007.

[19] M. A. Farries and A. L. Fairhall. Reinforcement Learning With Modulated Spike Timing Dependent Synaptic Plasticity. *Journal of Neurophysiology*, 98(6):3648–3665, 2007.

[20] R. Legenstein, D. Pecevski, and W. Maass. A learning theory for reward-modulated spike-timing-dependent plasticity with application to biofeedback. *PLoS Computational Biology*, 4(10):e1000180, 2008.

[21] C. T. Law and J. I. Gold. Reinforcement learning can account for associative and perceptual learning on a visual-decision task. *Nature Neuroscience*, 12(5):655–663, 2009.

[22] M. Biehl. An exactly solvable model of unsupervised learning. *Europhysics Letters*, 25(5):391–396, 1994.

[23] W. M. Baum. On two types of deviation from the matching law: Bias and undermatching. *Journal of the Experimental Analysis of Behavior*, 22(1):231–242, 1974.

[24] B. Barbour, N. Brunel, V. Hakim, and J. P. Nadal. What can we learn from synaptic weight distributions? *TRENDS in Neurosciences*, 30(12):622–629, 2007.

